# Modelling spatial recall, mental imagery and neglect

**Suzanna Becker**
Department of Psychology
McMaster University
1280 Main Street West
Hamilton, Ont. Canada L8S 4K1
*becker@mcmaster.ca*

**Neil Burgess**
Department of Anatomy and
Institute of Cognitive Neuroscience, UCL
17 Queen Square
London, UK WC1N 3AR
*n.burgess@ucl.ac.uk*

## Abstract

We present a computational model of the neural mechanisms in the parietal and temporal lobes that support spatial navigation, recall of scenes and imagery of the products of recall. Long term representations are stored in the hippocampus, and are associated with local spatial and object-related features in the parahippocampal region. Viewer-centered representations are dynamically generated from long term memory in the parietal part of the model. The model thereby simulates recall and imagery of locations and objects in complex environments. After parietal damage, the model exhibits hemispatial neglect in mental imagery that rotates with the imagined perspective of the observer, as in the famous Milan Square experiment [1]. Our model makes novel predictions for the neural representations in the parahippocampal and parietal regions and for behavior in healthy volunteers and neuropsychological patients.

## 1 Introduction

We perform spatial computations everday. Tasks such as reaching and navigating around visible obstacles are predominantly sensory-driven rather than memory-based, and presumably rely upon *egocentric*, or viewer-centered representations of space. These representations, and the ability to translate between them, have been accounted for in several computational models of the parietal cortex e.g. [2, 3]. In other situations such as route planning, recall and imagery for scenes or events one must also reply upon representations of spatial layouts from long-term memory. Neuropsychological and neuroimaging studies implicate both the parietal and hippocampal regions in such tasks [4, 5], with the long-term memory component associated with the hippocampus. The discovery of "place cells" in the hippocampus [6] provides evidence that hippocampal representations are *allocentric*, in that absolute locations in open spaces are encoded irrespective of viewing direction.

This paper addresses the nature and source of the spatial representations in the hippocampal and parietal regions, and how they interact during recall and navigation. We assume that in the hippocampus proper, long-term spatial memories are stored allocentrically, whereas in the parietal cortex view-based images are created on-the-fly during perception or recall. Intuitively it makes sense to use an allocentric representation for long-term storage as the

position of the body will have changed before recall. Alternatively, to act on a spatial location (e.g. reach with the hand) or to imagine a scene, an egocentric representation (e.g. relative to the hand or retina) is more useful [7, 8].

A study of hemispatial neglect patients throws some light on the interaction of long-term memory with mental imagery. Bisiach and Luzatti [1] asked two patients to recall the buildings from the familiar Cathedral Square in Milan, after being asked to imagine (i) facing the cathedral, and (ii) facing in the opposite direction. Both patients, in both (i) and (ii), predominantly recalled buildings that would have appeared on their right from the specified viewpoint. Since the buildings recalled in (i) were located physically on the opposite side of the square to those recalled in (ii), the patients' long-term memory for all of the buildings in the square was apparently intact. Further, the area neglected rotated according to the patient's imagined viewpoint, suggesting that their impairment relates to the generation of egocentric mental images from a non-egocentric long-term store.

The model also addresses how information about object identity is bound to locations in space in long-term memory, i.e. how the "what" and the "where" pathways interact. Object information from the ventral visual processing stream enters the hippocampal formation (medial entorhinal cortex) via the perirhinal cortex, while visuospatial information from the dorsal pathways enters lateral entorhinal cortex primarily via the parahippocampal cortex [9]. We extend the O'Keefe & Burgess [10] hippocampal model to include object-place associations by encoding object features in perirhinal cortex (we refer to these features as texture, but they could also be attributes such as colour, shape or size). Reciprocal connections to the parahippocampus allow object features to cue the hippocampus to activate a remembered location in an environment, and conversely, a remembered location can be used to reactivate the feature information of objects at that location. The connections from parietal to parahippocampal areas allow the remembered location to be specified in egocentric imagery.

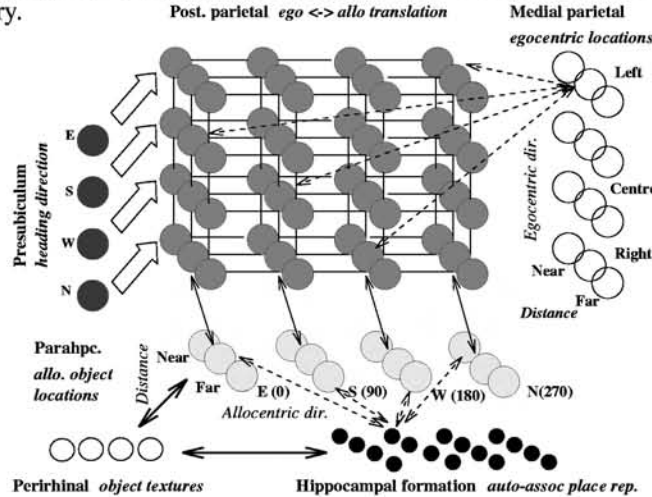

Figure 1: The model architecture. Note the allocentric encoding of direction (NSEW) in parahippocampus, and the egocentric encoding of directions (LR) in medial parietal cortex.

## 2  The model

The model may be thought of in simple terms as follows. An allocentric representation of object location is extracted from the ventral visual stream in the parahippocampus, and feeds into the hippocampus. The dorsal visual stream provides an egocentric representation of object location in medial parietal areas and makes bi-directional contact with the

parahippocampus via posterior parietal area 7a. Inputs carrying allocentric heading direction information [11] project to both parietal and parahippocampal regions, allowing bidirectional translation from allocentric to egocentric directions. Recurrent connections in the hippocampus allow recall from long-term memory via the parahippocampus, and egocentric imagery in the medial parietal areas. We now describe the model in more detail.

## 2.1 Hippocampal system

The architecture of the model is shown in Figure 1. The hippocampal formation (HF) consists of several regions – the entorhinal cortex, dentate gyrus, CA3, and CA1, each of which appears to code for space with varying degrees of sparseness. To simplify, in our model the HF is represented by a single layer of "place cells", each tuned to random, fixed configurations of spatial features as in [10, 12]. Additionally, it learns to represent objects' textural features associated with a particular location in the environment. It receives these inputs from the parahippocampal cortex (PH) and perirhinal cortex (PR), respectively.

The parahippocampal representation of object locations is simulated as a layer of neurons, each of which is tuned to respond whenever there is a landmark at a given distance and allocentric direction from the subject. Projections from this representation into the hippocampus drive the firing of place cells. This representation has been shown to account for the properties of place cells recorded across environments of varying shape and size [10, 12]. Recurrent connections between place cells allow subsequent pattern completion in the place cell layer. Return projections from the place cells to the parahippocampus allow reactivation of all landmark location information consistent with the current location.

The perirhinal representation in our model consists of a layer of neurons, each tuned to a particular textural feature. This region is reciprocally connected with the hippocampal formation [13]. Thus, in our model, object features can be used to cue the hippocampal system to activate a remembered location in an environment, and conversely, a remembered location can activate all associated object textures. Further, each allocentric spatial feature unit in the parahippocampus projects to the perirhinal object feature units so that attention to one location can activate a particular object's features.

## 2.2 Parietal cortex

Neurons responding to specific egocentric stimulus locations (e.g. relative to the eye, head or hand) have been recorded in several parietal areas. Tasks involving imagery of the products of retrieval tend to activate medial parietal areas (precuneus, posterior cingulate, retrosplenial cortex) in neuroimaging studies [14]. We hypothesize that there is a medial parietal egocentric map of space, coding for the locations of objects organised by distance and angle from the body midline. In this representation cells are tuned to respond to the presence of an object at a specific distance in a specific egocentric direction. Cells have also been reported in posterior parietal areas with egocentrically tuned responses that are modulated by variables such as eye position [15] or body orientation (in area 7a [16]). Such coding can allow translation of locations between reference frames [17, 2]. We hypothesize that area 7a performs the translation between allocentric and egocentric representations so that, as well as being driven directly by perception, the medial parietal egocentric map can be driven by recalled allocentric parahippocampal representations. We consider simply translation between allocentric and view-dependent representations, requiring a modulatory input from the head direction system. A more detailed model would include translations between allocentric and body, head and eye centered representations, and possibly use of retrosplenial areas to buffer these intermediate representations [18].

The translation between parahippocampal and parietal representations occurs via a hardwired mapping of each to an expanded set of egocentric representations, each modulated

by head direction so that one is fully activated for each (coarse coded) head direction (see Figure 1). With activation from the appropriate head direction unit, activation from the parahippocampal or parietal representation can activate the appropriate cell in the other representation via this expanded representation.

## 2.3 Simulation details

The hippocampal component of the model was trained on the spatial environment shown in the top-left panel of Figure 2, representing the buildings of the Milan square. We generated a series of views of the square, as would be seen from the locations in the central filled rectangular region of this figure panel. The weights were determined as follows, in order to form a continuous attractor (after [19, 20]). From each training location, each visible edge point contributed the following to the activation of each parahippocampal (PH) cell:

$$A_i^{PH} = \sum_j \sqrt{2\pi\sigma_{ang}^2}^{-1} e^{-\frac{(\theta_i - \theta_j)^2}{2\sigma_{ang}^2}} \times \sqrt{2\pi\sigma_{dir}(r_j)^2}^{-1} e^{-\frac{(r_i - r_j)^2}{2\sigma_{dir}(r_j)^2}} \quad (1)$$

where $\theta_i$ and $r_i$ are the preferred object direction and distance of the $i$th PH cell, $\theta_j$ and $r_j$ represent the location of the jth edge point relative to the observer, and $\sigma_{ang}$ and $\sigma_{dir}(r)$ are the corresponding standard deviations (as in [10]). Here, we used $\sigma_{ang} = pi/48$ and $\sigma_{dir}(r) = 2(r/10)^2$. The HF place cells were preassigned to cover a grid of locations in the environment, with each cell's activation falling off as a Gaussian of the distance to its preferred location. The PH-HF and HF-PH connection strengths were set equal to the correlations between activations in the parahippocampal and hippocampal regions across all training locations, and similarly, the HF-HF weights were set to values proportional to a Gaussian of the distance between their preferred locations.

The weights to the perirhinal (PR) object feature units – on the HF-to-PR and PH-to-PR connections – were trained by simulating sequential attention to each visible object, from each training location. Thus, a single object's textural features in the PR layer were associated with the corresponding PH location features and HF place cell activations via Hebbian learning. The PR-to-HF weights were trained to associate each training location with the single predominant texture – either that of a nearby object or that of the background.

The connections to and within the parietal component of the model were hard-wired to implement the bidirectional allocentric-egocentric mappings (these are functionally equivalent to a rotation by adding or subtracting the heading angle). The 2-layer parietal circuit in Figure 1 essentially encodes separate transformation matrices for each of a discrete set of head directions in the first layer. A right parietal lesion causing left neglect was simulated with graded, random knockout to units in the egocentric map of the left side of space. This could have equally been made to the trasnlation units projecting to them (i.e. those in the top rows of the PP in Figure 1).

After pretraining the model, we performed two sets of simulations. In simulation 1, the model was required to recall the allocentric representation of the Milan square after being cued with the texture and direction ($\theta_j$) of each of the visible buildings in turn, at a short distance $r_j$. The initial input to the HF, $I^{HF}(t = 0)$, was the sum of an externally provided texture cue from the PR cell layer, and a distance and direction cue from the PH cell layer obtained by initializing the PH states using equation 1, with $r_j = 2$. A place was then recalled by repeatedly updating the HF cells' states until convergence according to:

$$I^{HF}(t) = .25 I^{HF}(t-1) + .75 \left( W^{HF-HF} A^{HF}(t-1) + I^{HF}(0) \right) \quad (2)$$

$$A_i^{HF}(t) = exp(I_i^{HF}(t)) / \sum_k exp(I_k^{HF}(t)) \quad (3)$$

$$I^{PH}(t) = .9 I^{PH}(t-1) + .1 W^{HF-PH} A^{HF}(t) \quad (4)$$

Finally, the HF place cell activity was used to perform pattern completion in the PH layer (using the $W^{HF-PH}$ weights), to recall the other visible building locations. In simulation 2 the model was then required to generate view-based mental images of the Milan square from various viewpoints according to a specified heading direction. First, the PH cells and HF place cells were initialized to the states of the retrieved spatial location (obtained after settling in simulation 1). The model was then asked what it "saw" in various directions by simulating focused attention on the egocentric map, and requiring the model to retrieve the object texture at that location via activation of the PR region. The egocentric medial parietal (MP) activation was calculated from the PH-to-MP mapping, as described above. Attention to a queried egocentric direction was simulated by modulating the pattern of activation across the MP layer with a Gaussian filter centered on that location. This activation was then mapped back to the PH layer, and in turn projected to the PR layer via the PH-to-PR connections:

$$I^{PR} = W^{HC-PR}A^{HF} + W^{PH-PR}A^{PH} \tag{5}$$

$$A_i^{PR} = exp(I_i^{PR})/\sum_k exp(I_k^{PR}) \tag{6}$$

### 2.4   Results and discussion

In simulation 1, when cued with the textures of each of the 5 buildings around the training region, the model settled on an appropriate place cell activation. One such example is shown in Figure 2, upper panel. The model was cued with the texture of the cathedral front, and settled to a place representation near to its southwest corner. The resulting PH layer activations show correct recall of the locations of the other landmarks around the square. In simulation 2, shown in the lower panel, the model rotated the PH map according to the cued heading direction, and was able to retrieve correctly the texture of each building when queried with its egocentric direction. In the lesioned model, buildings to the egocentric left were usually not identified correctly. One such example is shown in Figure 2. The heading direction is to the south, so building 6 is represented at the top (egocentric forward) of the map. The building to the left has texture 5, and the building to the right has texture 7. After a simulated parietal lesion, the model neglects building 5.

## 3   Predictions and future directions

We have demonstrated how egocentric spatial representations may be formed from allocentric ones and vice versa. How might these representations and the mapping between them be learned? The entorhinal cortex (EC) is the major cortical input zone to the hippocampus, and both the parahippocampal and perirhinal regions project to it [13]. Single cell recordings in EC indicate tuning curves that are broadly similar to those of place cells, but are much more coarsely tuned and less specific to individual episodes [21, 9]. Additionally, EC cells can hold state information, such as a spatial location or object identity, over long time delays and even across intervening items [9]. An allocentric representation could emerge if the EC is under pressure to use a more compressed, temporally stable code to reconstruct the rapidly changing visuospatial input. An egocentric map is altered dramatically after changes in viewpoint, whereas an allocentric map is not. Thus, the PH and hippocampal representations could evolve via an unsupervised learning procedure that discovers a temporally stable, generative model of the parietal input. The inverse mapping from allocentric PH features to egocentric parietal features could be learned by training the back-projections similarly. But how could the egocentric map in the parietal region be learned in the first place? In a manner analogous to that suggested by Abbott [22], a "hidden layer" trained by Hebbian learning could develop egocentric features in learning a mapping from a sensory layer representing retinally located targets and arbitrary heading directions to a motor layer representing randomly explored (whole-body) movement directions.

We note that our parietal imagery system might also support the short-term visuospatial working memory required in more perceptual tasks (e.g. line cancellation)[2]. Thus lesions here would produce the commonly observed pattern of combined perceptual and representational neglect. However, the difference in the routes by which perceptual and reconstructed information would enter this system, and possibly in how they are manipulated, allow for patients showing only one form of neglect[23].

So far our simulations have involved a single spatial environment. Place cells recorded from the same rat placed in two similar novel environments show highly similar firing fields [10, 24], whereas after further exposure, distinctive responses emerge (e.g., [25, 26, 24] and unpublished data). In our model, sparse random connections from the object layer to the place layer ensure a high degree of initial place-tuning that should generalize across similar environments. Plasticity in the HF-PR connections will allow unique textures of walls, buildings etc to be associated with particular places; thus after extensive exposure, environment-specific place firing patterns should emerge.

A selective lesion to the parahippocampus should abolish the ability to make allocentric object-place associations altogether, thereby severely disrupting both landmark-based and memory-based navigation. In contrast, a pure hippocampal lesion would spare the ability to represent a single object's distance and allocentric directions from a location, so navigation based on a single landmark should be spared. If an arrangement of objects is viewed in a 3-D environment, the recall or recognition of the arrangement from a new viewpoint will be facilitated by having formed an allocentric representation of their locations. Thus we would predict that damage to the hippocampus would impair performance on this aspect of the task, while memory for the individual objects would be unimpaired. Similarly, we would expect a viewpoint-dependent effect in hemispatial neglect patients.

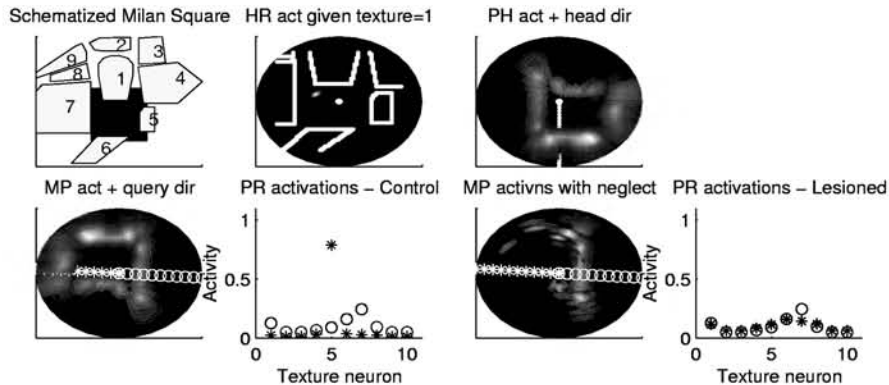

Figure 2: I. Top panel. Left: training locations in the Milan square are plotted in the black rectangle. Middle: HF place cell activations, after being cued that building #1 is nearby and to the north. Place cells are arranged in a polar coordinate grid according to the distance and direction of their preferred locations relative to the centre of the environment (bright white spot). The white blurry spot below and at the left end of building #1 is the maximally activated location. Edge points of buildings used during training are also shown here. Right: PH inputs to place cell layer are plotted in polar coordinates, representing the recalled distances and directions of visible edges associated with the maximally activated location. The externally cued heading direction is also shown here. II. Bottom panel. Left: An imagined view in the egocentric map layer (MP), given that the heading direction is south; the visible edges shown above have been rotated by 180 degrees. Mid-left: the recalled texture features in the PR layer are plotted in two different conditions, simulating attention to the right (circles) and left (stars). Mid-right and right: Similarly, the MP and PR activations are shown after damage to the left side of the egocentric map.

One of the many curiosities of the hemispatial neglect syndrome is the temporary amelioration of spatial neglect after left-sided vestibular stimulation (placement of cold water into the ear) and transcutaneous mechanical vibration (for a review, see [27]), which presumably affects the perceived head orientation. If the stimulus is evoking erroneous vestibular or somatosensory inputs to shift the perceived head direction system leftward, then all objects will now be mapped further rightward in egocentric space and into the 'good side' of the parietal map in a lesioned model. The model predicts that this effect will also be observed in imagery, as is consistent with a recent result [28].

### Acknowledgments

We thank Allen Cheung for extensive pilot simulations and John O'Keefe for useful discussions. NB is a Royal Society University Research Fellow. This work was supported by research grants from NSERC, Canada to S.B. and from the MRC, GB to N.B.

## References

[1] E. Bisiach and C. Luzzatti. *Cortex*, 14:129–133, 1978.

[2] A. Pouget and T. J. Sejnowski. *J. Cog. Neuro.*, 9(2):222–237, 1997.

[3] E. Salinas and L.F. Abbott. *J. Neurosci.*, 15:6461–6474, 1995.

[4] E.A. Maguire, N. Burgess, J.G. Donnett, R. S.J. Frackowiak, C.D. Frith, and J. O'Keefe. *Science*, 280:921–924, May 8 1998.

[5] N. Burgess, H. Spiers, E. Maguire, S. Baxendale, F. Vargha-Khadem, and J. O'Keefe. Subm.

[6] J. O'Keefe. *Exp. Neurol.*, 51:78–109, 1976.

[7] N. Burgess, K. Jeffery, and J. O'Keefe. In K.J. Jeffery N. Burgess and J. O'Keefe, editors, *The hippocampal and parietal foundations of spatial cognition*. Oxford U. Press, 1999.

[8] A.D. Milner, H.C. Dijkerman, and D.P. Carey. In K.J. Jeffery N. Burgess and J. O'Keefe, editors, *The hippocampal and parietal foundations of spatial cognition*. Oxford U. Press, 1999.

[9] W.A. Suzuki, E.K. Miller, and R. Desimone. *J. Neurosci.*, 78:1062–1081, 1997.

[10] J. O'Keefe and N. Burgess. *Nature*, 381:425–428, 1996.

[11] J.S. Taube. *Prog. Neurobiol.*, 55:225–256, 1998.

[12] T. Hartley, N. Burgess, C. Lever, F. Cacucci, and J. O'Keefe. *Hippocampus*, 10:369–379, 2000.

[13] W.A. Suzuki and D.G. Amaral. *J. Neurosci.*, 14:1856–1877, 1994.

[14] P.C. Fletcher, C.D. Frith, S.C. Baker, T. Shallice, R.S.J. Frackowiak, and R.J. Dolan. *Neuroimage*, 2(3):195–200, 1995.

[15] R.A. Andersen, G.K. Essick, and R.M. Siegel. *Science*, 230(4724):456–458, 1985.

[16] L.H. Snyder, A.P. Batista, and R.A. Andersen. *Nature*, 386:167–170, 1997.

[17] D. Zipser and R. A. Andersen. *Nature*, 331:679–684, 1988.

[18] N. Burgess, E. Maguire, H. Spiers, and J. O'Keefe. Submitted.

[19] A. Samsonovich and B.L. McNaughton. *J. Neurosci.*, 17:5900–5920, 1997.

[20] S. Deneve, P.E. Latham, and A. Pouget. *Nature Neuroscience*, 2(8):740–745, 1999.

[21] G.J. Quirk, R.U. Muller, J.L. Kubie, and J.B. Ranck. *J Neurosci*, 12:1945–1963, 1992.

[22] L.F. Abbott. *Int. J. of Neur. Sys.*, 6:115–122, 1995.

[23] C. Guariglia, A. Padovani, P. Pantano, and L. Pizzamiglio. *Nature*, 364:235–7, 1993.

[24] C. Lever, F. Cacucci, N. Burgess, and J. O'Keefe. In *Soc. Neurosci. Abs., vol. 24.*, 1999.

[25] E. Bostock, R.U. Muller, and J.L. Kubie. *Hippo.*, 1:193–205, 1991.

[26] R.U. Muller and J.L. Kubie. *J. Neurosci*, 7:1951–1968, 1987.

[27] G. Vallar. In K.J. Jeffery N. Burgess and J. O'Keefe, editors, *The hippocampal and parietal foundations of spatial cognition*. Oxford U. Press, 1999.

[28] C. Guariglia, G. Lippolis, and L. Pizzamiglio. *Cortex*, 34(2):233–241, 1998.
